# Semi-parametric exponential family PCA

**Sajama**          **Alon Orlitsky**
Department of Electrical and Computer Engineering
University of California at San Diego, La Jolla, CA 92093
`sajama@ucsd.edu` and `alon@ece.ucsd.edu`

## Abstract

We present a semi-parametric latent variable model based technique for density modelling, dimensionality reduction and visualization. Unlike previous methods, we estimate the latent distribution non-parametrically which enables us to model data generated by an underlying low dimensional, multimodal distribution. In addition, we allow the components of latent variable models to be drawn from the exponential family which makes the method suitable for special data types, for example binary or count data. Simulations on real valued, binary and count data show favorable comparison to other related schemes both in terms of separating different populations and generalization to unseen samples.

## 1 Introduction

Principal component analysis (PCA) is widely used for dimensionality reduction with applications ranging from pattern recognition and time series prediction to visualization. One important limitation of PCA is that it is not based on a probability model. A probabilistic formulation of PCA can offer several advantages like allowing statistical testing, application of Bayesian inference methods and naturally accommodating missing values [1]. Latent variable models are commonly used in statistics to summarize observations [2]. A latent variable model assumes that the distribution of data is determined by a latent or mixing distribution $P(\boldsymbol{\theta})$ and a conditional or component distribution $P(\mathbf{x}|\boldsymbol{\theta})$, i.e., $P(\mathbf{x}) = \int P(\boldsymbol{\theta})P(\mathbf{x}|\boldsymbol{\theta})d\boldsymbol{\theta}$.

Probabilistic PCA (PPCA) [1] borrows from one such popular model called factor analysis to propose a probabilistic alternative PCA. A key feature of this probabilistic model is that the latent distribution $P(\boldsymbol{\theta})$ is also assumed to be Gaussian since it leads to simple and fast model estimation, i.e., the density of $\mathbf{x}$ is approximated by a Gaussian distribution whose covariance matrix is aligned along a lower dimensional subspace. This may be a good approximation when data is drawn from a single population and the goal is to explain the data in terms of a few variables. However, in machine learning we often deal with data drawn from several populations and PCA is used to reduce dimensions to control computational complexity of learning. A mixture model with Gaussian latent distribution would not be able to capture this information. The projection obtained using a Gaussian latent distribution tends to be skewed toward the center [1] and hence the distinction between nearby sub-populations may be lost in the visualization space. For these reasons, it is important not to make restrictive assumptions about the latent distribution. Several recently proposed dimension reduction methods can, like PPCA, be thought of as special

cases of latent variable modelling which differ in the specific assumptions they make about the latent and conditional distributions.

We present an alternative probabilistic formulation, called semi-parametric PCA (SP-PCA), where no assumptions are made about the distribution of the latent random variable $\boldsymbol{\theta}$. Non-parametric latent distribution estimation allows us to approximate data density better than previous schemes and hence gives better low dimensional representations. In particular, multi-modality of the high dimensional density is better preserved in the projected space. When the observed data is composed of several clusters, this technique can be viewed as performing simultaneous clustering and dimensionality reduction. To make our method suitable for special data types, we allow the conditional distribution $P(\mathbf{x}|\boldsymbol{\theta})$ to be any member of the exponential family of distributions. Use of exponential family distributions for $P(\mathbf{x}|\boldsymbol{\theta})$ is common in statistics where it is known as latent trait analysis and they have also been used in several recently proposed dimensionality reduction schemes [3, 4]. We use Lindsay's non-parametric maximum likelihood estimation theorem to reduce the estimation problem to one with a large enough discrete prior. It turns out that this choice gives us a prior which is 'conjugate' to all exponential family distributions, allowing us to give a unified algorithm for all data types. This choice also makes it possible to efficiently estimate the model even in the case when different components of the data vector are of different types.

## 2   The constrained mixture model

We assume that the $d$-dimensional observation vectors $\mathbf{x}_1, \ldots, \mathbf{x}_n$ are outcomes of iid draws of a random variable whose distribution $P(\mathbf{x}) = \int P(\boldsymbol{\theta}) P(\mathbf{x}|\boldsymbol{\theta}) d\boldsymbol{\theta}$ is determined by the latent distribution $P(\boldsymbol{\theta})$ and the conditional distribution $P(\mathbf{x}|\boldsymbol{\theta})$. This can also be viewed as a mixture density with $P(\boldsymbol{\theta})$ being the *mixing distribution*, the mixture components labelled by $\boldsymbol{\theta}$ and $P(\mathbf{x}|\boldsymbol{\theta})$ being the *component distribution* corresponding to $\boldsymbol{\theta}$. The latent distribution is used to model the interdependencies among the components of $\mathbf{x}$ and the conditional distribution to model 'noise'. For example in the case of a collection of documents we can think of the 'content' of the document as a latent variable since it cannot be measured. For any given content, the words used in the document and their frequency may depend on random factors - for example what the author has been reading recently, and this can be modelled by $P(\mathbf{x}|\boldsymbol{\theta})$.

**Conditional distribution** $P(\mathbf{x}|\boldsymbol{\theta})$:    We assume that $P(\boldsymbol{\theta})$ adequately models the dependencies among the components of $\mathbf{x}$ and hence that the components of $\mathbf{x}$ are independent when conditioned upon $\boldsymbol{\theta}$, i.e., $P(\mathbf{x}|\boldsymbol{\theta}) = \Pi_j P(x_j|\theta_j)$, where $x_j$ and $\theta_j$ are the $j$'th components of $\mathbf{x}$ and $\boldsymbol{\theta}$. As noted in the introduction, using Gaussian means and constraining them to a lower dimensional subspace of the data space is equivalent to using Euclidean distance as a measure of similarity. This Gaussian model may not be appropriate for other data types, for instance the Bernoulli distribution may be better for binary data and Poisson for integer data. These three distributions, along with several others, belong to a family of distributions known as the *exponential family* [5]. Any member of this family can be written in the form

$$\log P(x|\theta) = \log P_0(x) + x\theta - G(\theta)$$

where $\theta$ is called the *natural parameter* and $G(\theta)$ is a function that ensures that the probabilities sum to one. An important property of this family is that the mean $\mu$ of a distribution and its natural parameter $\theta$ are related through a monotone invertible, nonlinear function $\mu = G'(\theta) = g(\theta)$. It can be shown that the negative log-likelihoods of exponential family distributions can be written as Bregman distances (ignoring constants) which are a family of generalized metrics associated with convex functions [4]. Note that by using different distributions for the various components of $\mathbf{x}$, we can model mixed data types.

**Latent distribution** $P(\boldsymbol{\theta})$: Like previous latent variable methods, including PCA, we constrain the latent variable $\boldsymbol{\theta}$ to an $\ell$-dimensional Euclidean subspace of $\mathbb{R}^d$ to model the belief that the intrinsic dimensionality of the data is smaller than $d$. One way to represent the (unknown) linear constraint on values that $\boldsymbol{\theta}$ can take is to write it as an invertible linear transformation of another random variable which takes values $\mathbf{a} \in \mathbb{R}^\ell$,

$$\boldsymbol{\theta} = \mathbf{a}V + \mathbf{b}$$

where $V$ is an $\ell \times d$ rotation matrix and $\mathbf{b}$ is a d-dimensional displacement vector. Hence any distribution $P_\Theta(\boldsymbol{\theta})$ satisfying the low dimensional constraints can be represented using a triple $(P(\mathbf{a}), V, \mathbf{b})$, where $P(\mathbf{a})$ is a distribution over $\mathbb{R}^\ell$. Lindsay's mixture non-parametric maximum likelihood estimation (NPMLE) theorem states that for fixed $(V,\mathbf{b})$, the maximum likelihood (ML) estimate of $P(\mathbf{a})$ exists and is a *discrete* distribution with no more than $n$ distinct points of support [6]. Hence if ML is the chosen parameter estimation technique, the SP-PCA model can be assumed (without loss of generality) to be a con-strained finite mixture model with at most $n$ mixture components. The number of mixture components in the model, $n$, grows with the amount of data and we propose to use pruning to reduce the number of components during model estimation to help both in computational speed and model generalization. Finally, we note that instead of the natural parameter, any of its invertible transformations could have been constrained to a lower dimensional space. Choosing to linearly constrain the natural parameter affords us computational advantages similar to those available when we use the canonical link in generalized linear regression.

**Low dimensional representation**: There are several ways in which low dimensional representations can be obtained using the constrained mixture model. We would ideally like to represent a given observation $\mathbf{x}$ by the unknown $\boldsymbol{\theta}$ (or the corresponding $\mathbf{a}$ related to $\boldsymbol{\theta}$ by $\boldsymbol{\theta} = \mathbf{a}V + \mathbf{b}$) that generated it, since the conditional distribution $P(\mathbf{x}|\boldsymbol{\theta})$ is used to model random effects. However, the actual value of $\mathbf{a}$ is not known to us and all of our knowledge of $\mathbf{a}$ is contained in the posterior distribution $P(\mathbf{a}|\mathbf{x}) = P(\mathbf{a})P(\mathbf{x}|\mathbf{a})/P(\mathbf{x})$. Since $\mathbf{a}$ belongs to an $\ell$-dimensional space, any of its estimators like the *posterior mean* or mode (MAP estimate) can be used to represent $\mathbf{x}$ in $\ell$ dimensions. For presenting the simulation results in this paper, we use the posterior mean as the representation point. This representation has been used in other latent variable methods to get meaningful low dimensional views [1, 3]. Another method is to represent $\mathbf{x}$ by that point $\boldsymbol{\theta}$ on $(V, b)$ that is closest according to the appropriate Bregman distance (it can be shown that there is a unique such $\boldsymbol{\theta}_{opt}$ on the plane). This representation is a generalization of the standard Euclidean projection and was used in [4].

**The Gaussian case**: When the exponential family distribution chosen is Gaussian, the model is a mixture of $n$ spherical Gaussians all of whose means lie on a hyperplane in the data space. This can be thought of as a 'soft' version of PCA, i.e., Gaussian case of SP-PCA is related to PCA in the same manner as Gaussian mixture model is related to K-means. The use of arbitrary mixing distribution over the plane allows us to approximate arbitrary spread of data along the hyperplane. Use of fixed variance spherical Gaussians ensures that like PCA, the direction perpendicular to the plane $(V, b)$ is irrelevant in any metric involving relative values of likelihoods $P(\mathbf{x}|\boldsymbol{\theta}_k)$, including the posterior mean.

Consider the case when data density $P(\mathbf{x})$ belongs to our model space, i.e., it is specified by $\{A, V, b, \Pi, \sigma\}$ and let $D$ be any direction parallel to the plane $(V, b)$ along which the latent distribution $P(\boldsymbol{\theta})$ has non-zero variance. Since Gaussian noise with variance $\sigma$ is added to this latent distribution to obtain $P(\mathbf{x})$, variance of $P(\mathbf{x})$ along $D$ will be greater than $\sigma$. The variance of $P(\mathbf{x})$ along any direction perpendicular to $(V, b)$ will be exactly $\sigma$. Hence, PCA of $P(\mathbf{x})$ yields the subspace $(V, b)$ which is the same as that obtained using SP-PCA (this may not be true when $P(\mathbf{x})$ does not belong to our model space). We found that SP-PCA differs significantly from PPCA in the predictive power of the low-dimensional

density model (see Section 5).

## 3  Model estimation

**Algorithm for ML estimation**:    We present an EM algorithm for estimating parameters of a finite mixture model with the components constrained to an $\ell$-dimensional Euclidean subspace. We propose an iterative re-weighted least squares (IRLS) method for the maximization step along the lines of generalized linear model estimation. Use of weighted least squares does not guarantee monotone increase in data likelihood. To guarantee convergence of the algorithm, we can check the likelihood of data at the IRLS update and decrease step size if necessary. Let $\mathbf{x}_1, \ldots, \mathbf{x}_n$ be iid samples drawn from a d-dimensional density $P(\mathbf{x})$, c be the number of mixture components and let the mixing density be $\Pi = (\pi_1, \ldots, \pi_c)$. Associated with each mixture component (indexed by k) are parameter vectors $\boldsymbol{\theta}_k$ and $\mathbf{a}_k$ which are related by $\boldsymbol{\theta}_k = \mathbf{a}_k V + b$. In this section we will work with the assumption that all components of $\mathbf{x}$ correspond to the same exponential family for ease of notation. For each observed $\mathbf{x}_i$ there is an unobserved 'missing' variable $\mathbf{z}_i$ which is a c-dimensional binary vector whose k'th component is one if the k'th mixture component was the outcome in the i'th random draw and zero otherwise. If $\mathbf{y}_l$ is a vector, we use $y_{lm}$ to denote its m'th component. (Derivation of the algorithm is omitted for lack of space, for details please see [7]).

The E-step is identical to unconstrained finite mixture case,

$$\hat{z}_{ik} = E(z_{ik}) = \frac{\pi_k P(\mathbf{x}_i / \boldsymbol{\theta}_k)}{\sum_{m=1}^{c} \pi_m P(\mathbf{x}_i / \boldsymbol{\theta}_m)} \quad ; \quad \tilde{x}_{kj} = \frac{\sum_{i=1}^{n} \hat{z}_{ik} x_{ij}}{\sum_{i=1}^{n} \hat{z}_{ik}}$$

In the M-step we update $\Pi$, $V$, $b$, and $\mathbf{a}_k$ in the following manner

$$\pi_k = \frac{\sum_{i=1}^{n} \hat{z}_{ik}}{\sum_{i=1}^{n} \sum_{m=1}^{c} z_{im}} = \frac{\sum_{i=1}^{n} \hat{z}_{ik}}{n}$$

$\mathbf{a}_i$ is updated by adding $\delta \mathbf{a}_i$ calculated using

$$(V \Omega_i V') \delta \mathbf{a}_i = G R_i \quad ; \quad [\Omega_i]_{qq} = \frac{\partial g(\theta_{iq})}{\partial \theta_{iq}} \quad ; \quad [GR_i]_{l1} = \sum_{j=1}^{d} (\tilde{x}_{ij} - g(\theta_{ij})) V_{lj}$$

Here the function $g(\theta)$ is as defined in Section 2 and depends on the member of the exponential family that is being used. Each column of the matrix $V$, $\mathbf{v}_s$, is updated by adding $\delta \mathbf{v}_s$ calculated using

$$(A' \Omega_s A) \delta \mathbf{v}_s = G R_s \quad ; \quad [\Omega_s]_{kk} = \frac{\partial g(\theta_{ks})}{\partial \theta_{ks}} \quad ; \quad [GR_s]_{l1} = \sum_{k'=1}^{c} (\tilde{x}_{k's} - g(\theta_{k's})) A_{k'l}$$

Each component of vector $\mathbf{b}$, $b_s$, is updated by adding $\delta b_s$ calculated using

$$H_s \delta b_s = G R_s \quad ; \quad H_s = \sum_{k'=1}^{c} \frac{\partial g(\theta_{k's})}{\partial \theta_{k's}} \quad ; \quad GR_s = \sum_{k'=1}^{c} (\tilde{x}_{k's} - g(\theta_{k's}))$$

**Pruning the mixture components**:    Redundant mixture components can be pruned between the EM iterations in order to improve speed of the algorithm and generalization properties while retaining the full capability to approximate $P(\mathbf{x})$. We propose the following criteria for pruning

- Starved components : If $\pi_k < C_1$, then drop the $k$'th component

- Nearby components : If $\max_i |P(\mathbf{x}_i|\boldsymbol{\theta}_{k1}) - P(\mathbf{x}|\boldsymbol{\theta}_{k2})| < C_2$, then drop either $k1$'th or $k2$'th component

The value of $C_1$ should be $\Theta(1/n)$ since we want to measure how starved a component is based on what percentage of the data it is 'responsible' for. To measure the nearness of components we use the $\infty$-norm of the difference between probabilities the components assign to observations since we do not want to lose mixture components that are distinguished with respect to a small number of observation vectors. In the case of clustering this means that we do not ignore under-represented clusters. $C_2$ should be chosen to be a small constant, depending on how much pruning is desired.

**Convergence of the EM iterations and computational complexity**:    It is easy to verify that the SP-PCA model satisfies the continuity assumptions of Theorem 2, [8], and hence we can conclude that any limit point of the EM iterations is a stationary point of the log likelihood function. The computational complexity of the E-step is $\mathcal{O}(cdn)$ and of the M-step is $\mathcal{O}(cd\ell^2)$. For the Gaussian case, the E-step only takes $\mathcal{O}(c\ell n)$ since we only need to take into account the variation of data along the subspace given by current value of V (see Section 2). The most expensive step is computation of $P(\mathbf{x}_i|\boldsymbol{\theta}_j)$. The k-d tree data structure is often used to identify relevant mixture components to speed up this step.

**Model selection**:    While any of the standard model selection methods based on penalizing complexity could be used to choose $\ell$, an alternative method is to pick $\ell$ which minimizes a validation or bootstrap based estimate of the prediction error (negative log likelihood per sample). For the Gaussian case, a fast method to pick $\ell$ would be to plot the variance of data along the principal directions (found using PCA) and look for the dimension at which there is a 'knee' or a sudden drop in variance or where the total residual variance falls below a chosen threshold.

**Consistency of the Maximum Likelihood estimator**:    We propose to use the ML estimator to find the latent space $(V, b)$ and the latent distribution $P(\mathbf{a})$. Usually a parametric form is assumed for $P(\mathbf{a})$ and the consistency of the ML estimate is well known for this task where the parameter space is a subset of a finite dimensional Euclidean space. In the SP-PCA model, one of the parameters $(P(\mathbf{a}))$ ranges over the space of all distribution functions on $\mathbb{R}^\ell$ and hence we need to do more to verify the validity of our estimator. Exponential family mixtures are not identifiable in general. This, however, is not a problem for us since we are only interested in approximating $P(\mathbf{x})$ well and not in the actual parameters corresponding to the distribution. Hence we use the definition of consistency of an estimator given by Redner. Let $\gamma_0$ be the 'true' parameter from which observed samples are drawn. Let $C_0$ be the set of all parameters $\gamma$ corresponding to the 'true' distribution $F(x/\gamma_0)$ (i.e., $C_0 = \{\gamma : F(x/\gamma) = F(x/\gamma_0) \; \forall \; x\}$). Let $\hat{\gamma}_n$ be an estimator of $\gamma$ based on n observed samples of X and let $\hat{\Gamma}$ be the quotient topological space obtained from $\Gamma$ obtained by identifying the set $C_0$ to a point $\hat{\gamma}_0$.

**Definition**    *The sequence of estimators $\{\hat{\gamma}_n, \; n = 1, \ldots, \infty\}$ is said to be strongly consistent in the sense of Redner if $\lim_{m \to \infty} \hat{\gamma}_n = \hat{\gamma}_0$ almost surely.*

**Theorem**    *If $P(\mathbf{a})$ is assumed to be zero outside a bounded subset of $\mathbb{R}^\ell$, the ML estimator of parameter $(V, b, P(\mathbf{a}))$ is strongly consistent for Gaussian, Binary and Poisson conditional distributions.*

The theorem follows by verifying that the assumptions of Kiefer et. al. [9] are satisfied by the SP-PCA model. The assumption that $P(\mathbf{a})$ is zero outside a bounded region is not restrictive in practice since we expect the observations $\mathbf{x}_i$ belong to a bounded region of $\mathbb{R}^d$. (Proof omitted for lack of space, please see [7]).

Table 1: Bootstrap estimates of prediction error for PPCA and SP-PCA.

| DENSITY | ISOTROPIC GAUSSIAN | PPCA | | | SP-PCA | | | FULL GAUSSIAN |
|---|---|---|---|---|---|---|---|---|
| | | $\ell$=1 | $\ell$=2 | $\ell$=3 | $\ell$=1 | $\ell$=2 | $\ell$=3 | |
| ERROR | 50.39 | 38.03 | 34.71 | 34.76 | 36.85 | 30.99 | 28.54 | 343.83 |

## 4   Relationship to past work

SP-PCA is a factor model that makes fewer assumptions about latent distribution than PPCA [1]. Mixtures of probabilistic principal component analyzers (also known as mixtures of factor analyzers) is a generalization of PPCA which overcomes the limitation of global linearity of PCA via local dimensionality reduction. Mixtures of SP-PCA's can be similarly defined and used for local dimensionality reduction. Collins et. al. [4] proposed a generalization of PCA using exponential family distributions. Note that this generalization is not associated with a probability density model for the data. SP-PCA can be thought of as a 'soft' version of this generalization of PCA, in the same manner as Gaussian mixtures are a soft version of K-means. Generative topographic mapping (GTM) is a probabilistic alternative to Self organizing map which aims at finding a nonlinear lower dimensional manifold passing close to data points. An extension of GTM using exponential family distributions to deal with binary and count data is described in [3]. Apart from the fact that GTM is a non-linear dimensionality reduction technique while SP-PCA is globally linear like PCA, one main feature that distinguishes the two is the choice of latent distribution. GTM assumes that the latent distribution is uniform over a finite and discrete grid of points. Both the location of the grid and the nonlinear mapping are to be given as an input to the algorithm. Tibshirani [10] used a semi-parametric latent variable model for estimation of principle curves. Discussion of these and other dimensionality reduction schemes based on latent trait and latent class models can be found in [7].

## 5   Experiments

In this section we present simulations on synthetic and real data to demonstrate the properties of SP-PCA. In factor analysis literature, it is commonly believed that choice of prior distribution is unimportant for the low dimensional data summarization (see [2], Sections 2.3, 2.10 and 2.16). Through the examples below we argue that estimating the prior instead of assuming it arbitrarily can make a difference when latent variable models are used for density approximation, data analysis and visualization.

**Use of SP-PCA as a low dimensional density model**:     The Tobamovirus data which consists of 38 18-dimensional examples was used in [1] to illustrate properties of PPCA. PPCA and SP-PCA can be thought of as providing a range of low-dimensional density models for the data. The complexity of these densities increases with and is controlled by the value of $\ell$ (the projected space dimension) starting with the zero dimensional model of an isotropic Gaussian. For a fixed lower dimension $\ell$, SP-PCA has greater approximation capability than PPCA. In Table 1, we present bootstrap estimates of the predictive power of PPCA and SP-PCA for various values of L. SP-PCA has lower prediction error than PPCA for $\ell = 1, 2$ and 3. This indicates that SP-PCA combines flexible density estimation and excellent generalization even when trained on a small amount of data.

**Simulation results on discrete datasets**:     We present experiments on 20 Newsgroups dataset comparing SP-PCA to PCA, exponential family GTM [3] and Exponential family PCA [4]. Data for the first set of simulations was drawn from comp.sys.ibm.pc.hardware, comp.sys.mac.hardware and sci.med newsgroups. A dictionary size of 150 words was chosen and the words in the dictionary were picked to be those which have maximum mutual information with class labels. 200 documents were drawn from each of the three

newsgroups to form the training data. Two-dimensional representations obtained using various methods are shown in Fig. 1. In the projection obtained using PCA, Exponential family PCA and Bernoulli GTM, the classes comp.sys.ibm.pc.hardware and comp.sys.-mac.hardware were not well separated in the 2D space. This result (Fig. 1(c)) was presented in [3] and the the overlap between the two groups was attributed to the fact that they are very similar and hence share many words in common. However, SP-PCA was able to separate the three sets reasonably well (Fig. 1(d)). One way to quantify the separation of dissimilar groups in the two-dimensional projections is to use the training set classification error of projected data using SVM. The accuracy of the best SVM classifier (we tried a range of SVM parameter values and picked the best for each projected data set) was 75% for bernoulli GTM projection and 82.3% for SP-PCA projection (the difference corresponds to 44 data points while the total number of data points is 600). We conjecture that the reason comp.sys.ibm.pc.hardware and comp.sys.mac.hardware have overlap in projection using Bernoulli GTM is that the prior is assumed to be over a pre-specified grid in latent space and the spacing between grid points happened to be large in the parameter space close to the two news groups. In contrast to this, in SP-PCA there is no grid and the latent distribution is allowed to adapt to the given data set. Note that a standard clustering algorithm could be used on the data projected using SP-PCA to conclude that data consisted of three kinds of documents.

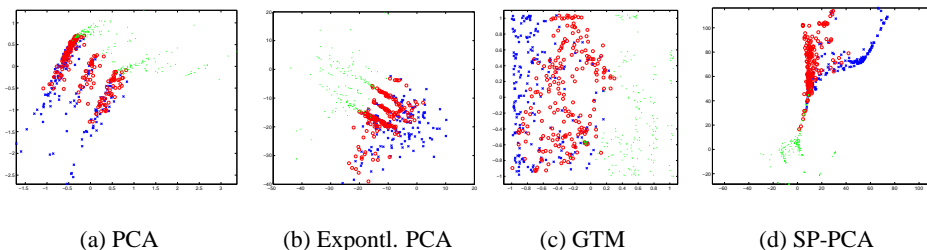

|          (a) PCA          |          (b) Expontl. PCA          |          (c) GTM          |          (d) SP-PCA          |

Figure 1: Projection by various methods of binary data from 200 documents each from comp.sys.ibm.pc.hardware ($\times$), comp.sys.mac.hardware ($\circ$) and sci.med (.)

Data for the second set of simulations was drawn from sci.crypt, sci.med, sci.space and soc.culture.religion.christianity newsgroups. A dictionary size of 100 words was chosen and again the words in the dictionary were picked to be those which have maximum mutual information with class labels. 100 documents were drawn from each of the newsgroups to form the training data and 100 more to form the test data. Fig. 2 shows two-dimensional representations of binary data obtained using various methods. Note that while the four newsgroups are bunched together in the projection obtained using Exponential family PCA [4] (Fig. 2(b)), we can still detect the presence four groups from this projection and in this sense this projection is better than the PCA projection. This result is pleasing since it confirms our intuition that using negative log-likelihood of Bernoulli distribution as a measure of similarity is more appropriate than squared Euclidean distance for binary data. We conjecture that the reason the four groups are not well separated in this projection is that a conjugate prior has to be used in its estimation for computational purposes [4] and the form and parameters of this prior are considered fixed and given inputs to the algorithm. Both SP-PCA (Fig. 2(c)) and Bernoulli GTM (Fig. 2(e)) were able to clearly separate the clusters in the training data. Figures 2(d) and 2(f) show representation of test data using the models estimated by SP-PCA and Bernoulli GTM respectively. To measure generalization of these methods, we use a K-nearest neighbors based non-parametric estimate of the density of the projected training data. The percentage difference between the log-likelihoods of training and test data with respect to this density was 9.1% for SP-PCA and 17.6% for GTM for K=40 (SP-PCA had smaller percentage change in log-likelihood for most values of K that we tried between 10 and 40). This indicates that SP-PCA generalizes better than

GTM. This can be seen visually by comparing Figures 2(e) and 2(f) where the projections of training and test data of sci.space ($\nabla$) differ significantly.

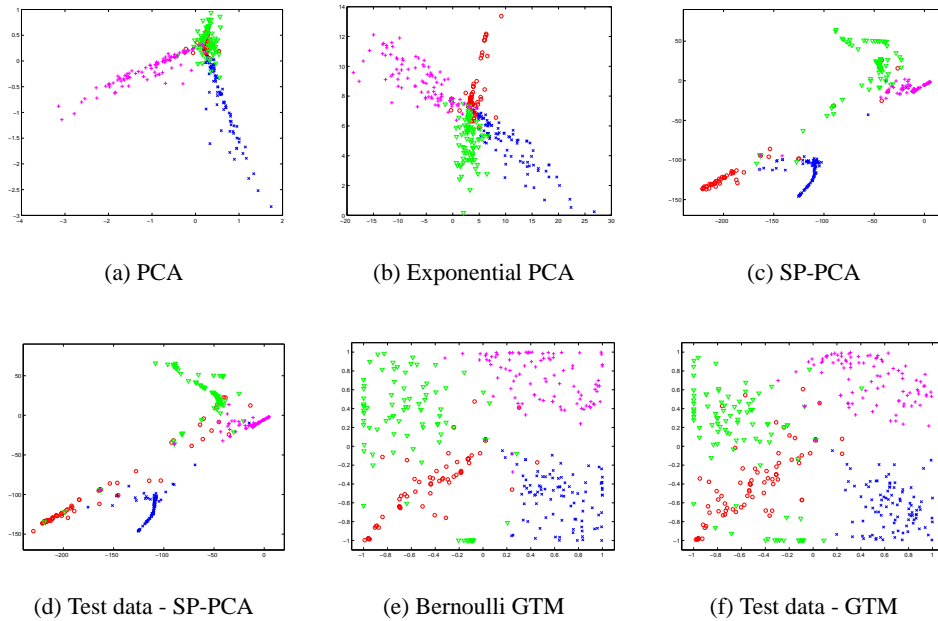

(a) PCA        (b) Exponential PCA        (c) SP-PCA

(d) Test data - SP-PCA        (e) Bernoulli GTM        (f) Test data - GTM

Figure 2: Projection by various methods of binary data from 100 documents each from sci.crypt ($\times$), sci.med ($\circ$), sci.space ($\nabla$) and soc.culture.religion.christianity ($+$)

## Acknowledgments

We thank Sanjoy Dasgupta and Thomas John for helpful conversations.

## References

[1] M. Tipping and C. Bishop. Probabilistic principal component analysis. *Journal of the Royal Statistical Society, Series B*, 61(3):611–622, 1999.

[2] David J. Bartholomew and Martin Knott. *Latent variable models and Factor analysis*, volume 7 of *Kendall's Library of Statistics*. Oxford University Press, 2nd edition, 1999.

[3] A. Kaban and M. Girolami. A combined latent class and trait model for the analysis and visualization of discrete data. *IEEE Transaction on Pattern Analysis and Machine Intelligence*, 23(8):859–872, August 2001.

[4] M. Collins, S. Dasgupta, and R. E. Schapire. A generalization of principal components analysis to the exponential family. In *Advances in Neural Information Processing Systems 14*, 2002.

[5] P. McCullagh and J. A. Nelder. *Generalized Linear Models*. Monographs on Statistics and Applied Probability. Chapman and Hall, 1983.

[6] B. G. Lindsay. The geometry of mixture likelihoods : A general theory. *The Annals of Statistics*, 11(1):86–04, 1983.

[7] Sajama and A. Orlitsky. Semi-parametric exponential family PCA : Reducing dimensions via non-parametric latent distribution estimation. Technical Report CS2004-0790, University of California at San Diego, http://cwc.ucsd.edu/$\sim$ sajama, 2004.

[8] C. F. J. Wu. On the convergence properties of the EM algorithm. *Annals of Statistics*, 11(1):95–103, 1983.

[9] J. Kiefer and J. Wolfowitz. Consistency of the maximum likelihood estimator in the presence of infinitely many incidental parameters. *The Annals of Mathematical Statistics*, 27:887–906, 1956.

[10] R. Tibshirani. Principal curves revisited. *Statistics and Computation*, 2:183–190, 1992.
